# The Infinite Markov Model

**Daichi Mochihashi** [*]
NTT Communication Science Laboratories
Hikaridai 2-4, Keihanna Science City
Kyoto, Japan 619-0237
daichi@cslab.kecl.ntt.co.jp

**Eiichiro Sumita**
ATR / NICT
Hikaridai 2-2, Keihanna Science City
Kyoto, Japan 619-0288
eiichiro.sumita@atr.jp

## Abstract

We present a nonparametric Bayesian method of estimating variable order Markov processes up to a theoretically infinite order. By extending a stick-breaking prior, which is usually defined on a unit interval, "vertically" to the trees of infinite depth associated with a hierarchical Chinese restaurant process, our model directly infers the hidden orders of Markov dependencies from which each symbol originated. Experiments on character and word sequences in natural language showed that the model has a comparative performance with an exponentially large full-order model, while computationally much efficient in both time and space. We expect that this basic model will also extend to the variable order hierarchical clustering of general data.

## 1 Introduction

Since the pioneering work of Shannon [1], Markov models have not only been taught in elementary information theory classes, but also served as indispensable tools and building blocks for sequence modeling in many fields, including natural language processing, bioinformatics [2], and compression [3]. In particular, $(n-1)$th order Markov models over words are called "n-gram" language models and play a key role in speech recognition and machine translation, as regards choosing the most natural sentence among candidate transcriptions [4].

Despite its mathematical simplicity, an inherent problem with a Markov model is that we must determine its order. Because higher-order Markov models have an exponentially large number of parameters, their orders have been restricted to a small, often fixed number. In fact, for "n-gram" models the assumed word dependency $n$ is usually set at from three to five due to the high dimensionality of the lexicon. However, word dependencies will often have a span of greater than $n$ for phrasal expressions or compound proper nouns, or a much shorter $n$ will suffice for some grammatical relationships. Similarly, DNA or amino acid sequences might have originated from multiple temporal scales that are unknown to us.

To alleviate this problem, many "variable-order" Markov models have been proposed [2, 5, 6, 7]. However, all stemming from [5] and [7], they are based on pruning a huge candidate suffix tree by employing such criteria as KL-divergences. This kind of "post-hoc" approach suffers from several important limitations: First, when we want to consider deeper dependences, the candidate tree to be pruned will be extremely large. This is especially prohibitive when the lexicon size is large as with language. Second, the criteria and threshold for pruning the tree are inherently exogenous and must be set carefully so that they match the desired model and current data. Third, pruning by empirical counts in advance, which is often used to build "arbitrary order" candidate trees in these approaches, is shown to behave very badly [8] and has no theoretical standpoints.

In contrast, in this paper we propose a complete generative model of variable-order Markov processes up to a theoretically infinite order. By extending a stick-breaking prior, which is usually

---

[*]This research was conducted while the first author was affiliated with ATR/NICT.

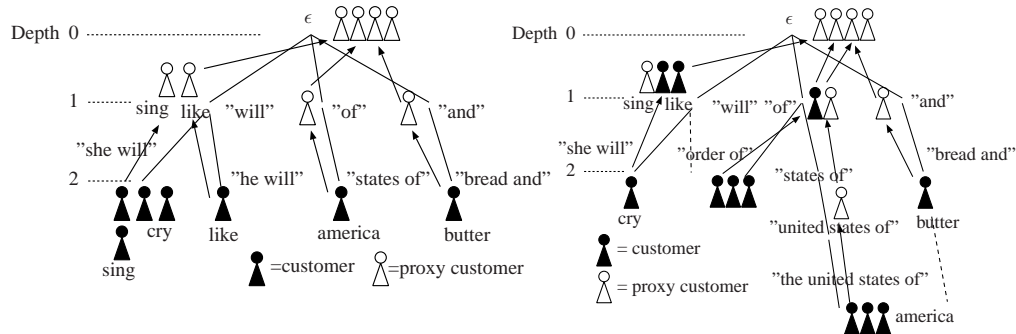

(a) Suffix Tree representation of the hierarchical Chinese Restaurant process on a second-order Markov model. Each count is a customer in this suffix tree.

(b) Infinite suffix tree of the proposed model. Deploying customers at suitable depths, i.e. Markov orders, is our inference problem.

Figure 1: Hierarchical Chinese restaurant processes over the finite and infinite suffix trees.

defined on a unit interval, "vertically" to the trees of infinite depth associated with a hierarchical Chinese restaurant process, our model directly infers the hidden orders of Markov dependencies from which each symbol originated. We show this is possible with a small change to the inference of the hiearchical Pitman-Yor process in discrete cases, and actually makes it more efficient in both computational time and space. Furthermore, we extend the variable model by latent topics to show that we can induce the variable length "stochastic phrases" for topic by topic.

## 2 Suffix Trees on Hierarchical Chinese Restaurant Processes

The main obstacle that has prevented consistent approaches to variable order Markov models is the lack of a hierarchical generative model of Markov processes that allows estimating increasingly sparse distributions as its order gets larger. However, now we have the hierarchical (Poisson-) Dirichlet process that can be used as a fixed order language model [9][10], it is natural for us to extend these models to variable orders also by using a nonparametric approach. While we concentrate here on discrete distributions, the same basic approach can be applied to a Markov process on continuous distributions, such as Gaussians that inherit their means from their parent distributions. For concreteness below we use a language model example, but the same model can be applied to any discrete sequences, such as characters, DNAs, or even binary streams for compression.

Consider a trigram language model, which is a second-order Markov model over words often employed in speech recognition. Following [9], this Markov model can be represented by a suffix tree of depth two, as shown in Figure 1(a). When we predict a word "sing" after a context "she will", we descend this suffix tree from the root (which corresponds to null string context), using the context backwards to follow a branch "will" and then "she will".[1] Now we arrive at the leaf node that represents the context, and we can predict "sing" by using the count distribution at this node.

During the learning phase, we begin with a suffix tree that has no counts. For every time a three word sequence appears in the training data, such as "she will sing" mentioned above, we add a count of a final word ("sing") given the context ("she will") to the context node in the suffix tree. In fact this corresponds to a hierarchical Chinese restaurant process, where each context node is a restaurant and each count is a customer associated with a word. Here each node, i.e. restaurant, might not have customers for all the words in the lexicon. Therefore, when a customer arrives at a node and stochastically needs a new table to sit down, a copy of him, namely a *proxy customer*, is sent to its parent node. When a node has no customer to compute the probability of some word, it uses the distribution of customers at the parent node and appropriately interpolates it to sum to 1.

Assume that the node "she will" does not have a customer of "like." We can nevertheless compute the probability of "like" given "she will" if its sibling "he will" has a customer "like". Because that sibling has sent a copy of the customer to the common parent "will", the probability is computed by appropriately interpolating the trigram probability given "she will", which is zero, with the bigram probability given "will", which is not zero at the parent node.

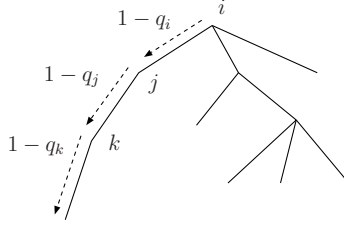

Figure 2: Probabilistic suffix tree of an infinite depth. $(1-q_i)$ is a "penetration probability" of a descending customer at each node $i$, defining a stick-breaking process over the infinite tree.

Consequently, in the hierarchical Pitman-Yor language model (HPYLM), the predictive probability of a symbol $s = s_t$ in context $h = s_{t-n} \cdots s_{t-1}$ is recursively computed by

$$p(s|h) = \frac{c(s|h) - d \cdot t_{hs}}{\theta + c(h)} + \frac{\theta + d \cdot t_{h\cdot}}{\theta + c(h)} \, p(s|h'),  \qquad (1)$$

where $h' = s_{t-n+1} \cdots s_{t-1}$ is a shortened context with the farthest symbol dropped. $c(s|h)$ is the count of $s$ at node $h$, and $c(h) = \sum_s c(s|h)$ is the total count at node $h$. $t_{hs}$ is the number of times symbol $s$ is estimated to be generated from its parent distribution $p(s|h')$ rather than $p(s|h)$ in the training data: $t_{h\cdot} = \sum_s t_{hs}$ is its total. $\theta$ and $d$ are the parameters of the Pitman-Yor process, and can be estimated through the distribution of customers on a suffix tree by Gamma and Beta posterior distributions, respectively. For details, see [9].

Although this Bayesian Markov model is very principled and attractive, we can see from Figure 1(a) that all the real customers (i.e., counts) are fixed at the depth $(n-1)$ in the suffix tree. Because actual sequences will have heterogeneous Markov dependencies, we want a Markov model that deploys customers at different levels in the suffix tree according to the true Markov order from which each customer originated. But how can we model such a heterogeneous property of Markov sequences?

## 3  Infinite-order Hierarchical Chinese Restaurant Processes

Intuitively, we know that suffix trees that are too deep are improbable and symbol dependencies decay largely exponentially with context lengths. However, some customers may reside in a very deep node (for example, "the united states of america") and some in a shallow node ("shorter than"). Our model for deploying customers must be flexible enough to accommodate all these possibilities.

### 3.1  Introducing Suffix Tree Prior

For this purpose, we assume that each node $i$ in the suffix tree has a *hidden* probability $q_i$ of stopping at node $i$ when following a path from the root of the tree to add a customer. In other words, $(1 - q_i)$ is the "penetration probability" when descending an infinite depth suffix tree from its root (Figure 2). We assume that each $q_i$ is generated from a prior Beta distribution independently as:

$$q_i \sim \mathrm{Be}(\alpha, \beta) \qquad \text{i.i.d.} \qquad (2)$$

This choice is mainly for simplicity: however, later we will show that the final predictive performance does not significantly depend on $\alpha$ or $\beta$.

When we want to generate a symbol $s_t$ given a context $h = s_{-\infty} \cdots s_{t-2} s_{t-1}$, we descend the suffix tree from the root following a path $s_{t-1} \to s_{t-2} \to \cdots$, according to the probability of stopping at a level $l$ given by

$$p(n = l|h) = q_l \prod_{i=0}^{l-1} (1 - q_i). \qquad (l = 0, 1, \cdots, \infty) \qquad (3)$$

When we stop at level $l$, we generate a symbol $s_t$ using the context $s_{t-l} \cdots s_{t-2} s_{t-1}$. Since $q_i$ differs from node to node, we may reach very deep nodes with high probability if the $q_i$'s along the path are equally small (the "penetration" of this branch is high); or, we may stop at a very shallow node if the $q_i$'s are very high (the "penetration" is low). In general, the probability to reach a node decays exponentially with levels according to (3), but the degrees are different to allow for long sequences of typical phrases.

Note that even for the same context $h$, the context length that was used to generate the next symbol may differ stochastically for each appearance according to (3).

## 3.2 Inference

Of course, we do not know the hidden probability $q_i$ possessed by each node. Then, how can we estimate it? Note that the generative model above amounts to introducing a vector of hidden variables, $\mathbf{n} = n_1 n_2 \cdots n_T$, that corresponds to each Markov order ($n = 0 \cdots \infty$) from which each symbol $s_t$ in $\mathbf{s} = s_1 s_2 \cdots s_T$ originated. Therefore, we can write the probability of $\mathbf{s}$ as follows:

$$p(\mathbf{s}) = \sum_{\mathbf{n}} \sum_{\mathbf{z}} p(\mathbf{s}, \mathbf{z}, \mathbf{n}) . \tag{4}$$

Here, $\mathbf{z} = z_1 z_2 \cdots z_T$ is a vector that represents the hidden seatings of the proxy customers described in Section 2, where $0 \leq z_t \leq n_t$ means how recursively the $s_t$'s proxy customers are stochastically sent to parent nodes. To estimate these hidden variables $\mathbf{n}$ and $\mathbf{z}$, we use a Gibbs sampler as in [9]. Since in the hierarchical (Poisson-)Dirichlet process the customers are exchangeable [9] and $q_i$ is i.i.d. as shown in (2), this process is also exchangeable and therefore we can always assume, by a suitable permutation, that the customer to resample is the final customer.

In our case, we only explicitly resample $n_t$ given $\mathbf{n}_{-t}$ ($\mathbf{n}$ excluding $n_t$), as follows:

$$n_t \sim p(n_t | \mathbf{s}, \mathbf{z}_{-t}, \mathbf{n}_{-t}). \tag{5}$$

Notice here that when we sample $n_t$, we already know the other depths $\mathbf{n}_{-t}$ that other words have reached in the suffix tree. Therefore, when computing (5) using (3), the expectation of each $q_i$ is

$$E[q_i] = \frac{a_i + \alpha}{a_i + b_i + \alpha + \beta} , \tag{6}$$

where $a_i$ is the number of times node $i$ was stopped at when generating other words, and $b_i$ is the number of times node $i$ was passed by. Using this estimate, we decompose the conditional probability of (5) as

$$p(n_t | \mathbf{s}, \mathbf{z}_{-t}, \mathbf{n}_{-t}) \propto p(s_t | \mathbf{s}_{-t}, \mathbf{z}_{-t}, \mathbf{n}) \, p(n_t | \mathbf{s}_{-t}, \mathbf{z}_{-t}, \mathbf{n}_{-t}) . \tag{7}$$

The first term is the probability of $s_t$ under HPYLM when the Markov order is known to be $n_t$, given by (1). The second term is the prior probability of reaching that node at depth $n_t$. By using (6) and (3), this probability is given by

$$p(n_t = l | \mathbf{s}_{-t}, \mathbf{z}_{-t}, \mathbf{n}_{-t}) = \frac{a_l + \alpha}{a_l + b_l + \alpha + \beta} \prod_{i=0}^{l-1} \frac{b_i + \beta}{a_i + b_i + \alpha + \beta} . \tag{8}$$

Expression (7) is a tradeoff between these two terms: the prediction of $s_t$ will be increasingly better when the context length $n_t$ becomes long, but we can select it only when the probability of reaching that level in the suffix tree is supported by the other counts in the training data.

Using these probabilities, we can construct a Gibbs sampler, as shown in Figure 3, to iteratively resample $\mathbf{n}$ and $\mathbf{z}$ in order to estimate the parameter of the variable order hierarchical Pitman-Yor language model (VPYLM)[2]. In this sampler, we first remove the $t$'th customer who resides at a depth of order[$t$] in the suffix tree, and decrement $a_i$ or $b_i$ accordingly along the path. Sampling a new depth (i.e. Markov order) according to (7), we put the $t$'th customer back at the new depth recorded as order[$t$], and increment $a_i$ or $b_i$ accordingly along the new path. When we add a customer $s_t$, $z_t$ is implicitly sampled because $s_t$'s proxy customer is recursively sent to parent nodes in case a new table is needed to sit him down.

```
1: for j = 1 ··· N do
2:    for t = randperm(1 ··· T) do
3:       if j > 1 then
4:          remove_customer (order[t], s_t, s_{1:t-1})
5:       end if
6:       order[t] = add_customer (s_t, s_{1:t-1}).
7:    end for
8: end for
```

Figure 3: Gibbs Sampler of VPYLM.

```
struct ngram {          /* n-gram node */
    ngram *parent;
    splay *children;    /* = (ngram **) */
    splay *symbols;     /* = (restaurant **) */
    int stop;           /* a_h */
    int through;        /* b_h */
    int ncounts;        /* c(h) */
    int ntables;        /* t_h. */
    int id;             /* symbol id */
};
```

Figure 4: Data structure of a suffix tree node. Counts $a_h$ and $b_h$ are maintained at each node. We used Splay Trees for efficient insertion/deletion.

> 'how queershaped little children drawling-desks, which would get through that dormouse!' said alice; 'let us all for anything the secondly, but it to have and another question, but i shalled out, 'you are old,' said the you're trying to far out to sea.

(a) Random walk generation from a character model.

| Character | s a i d ␣ a l i c e ; ␣ ' l e t ␣ u s ␣ a l l ␣ f o r ␣ a n y t h i n g ␣ t h e ␣ s e c o n d l y , ␣ ··· |
|---|---|
| Markov order | 5 6 5 4 7 1 0 6 5 4 3 7 1 4 8 2 4 4 6 5 5 4 4 5 5 6 4 5 6 7 7 7 5 3 3 4 5 9 1 1 6 4 8 9 8 9 4 4 4 7 3 4 3 ··· |

(b) Markov orders used to generate each character above.

Figure 5: Character-based infinite Markov model trained on "Alice in Wonderland."

This sampler is an extension of that reported in [9] using stochastically different orders $n$ ($n = 0 \cdots \infty$) for each customer. In practice, we can place some maximum order $n_{\max}$ on $n$ and sample within it [3], or use a small threshold $\epsilon$ to stop the descent when the prior probability (8) of reaching that level is smaller than $\epsilon$. In this case, we obtain an "infinite" order Markov model: now we can eliminate the order from Markov models by integrating it out.

Because each node in the suffix tree may have a huge number of children, we used Splay Trees [11] for the efficient search as in [6]. Splay Trees are self-organizing binary search trees having amortized $O(\log n)$ order, that automatically put frequent items at shallower nodes. This is ideal for sequences with a power law property like natural languages. Figure 4 shows our data structure of a node in a suffix tree.

### 3.3  Prediction

Since we do not usually know the Markov order of a context $h = s_{-\infty} \cdots s_{-2} s_{-1}$ beforehand, when making predictions we consider $n$ as a latent variable and average over it, as follows:

$$p(s|h) = \sum_{n=0}^{\infty} p(s, n|h) \tag{9}$$
$$= \sum_{n=0}^{\infty} p(s|h, n)p(n|h). \tag{10}$$

Here, $p(s|n, h)$ is a HPYLM prediction of order $n$ through (1), and $p(n|h)$ is the probability distribution of latent Markov order $n$ possessed by the context $h$, obtained through (8). In practice, we further average (10) over the configurations of $\mathbf{n}$ and $\mathbf{s}$ through $N$ Gibbs iterations on training data $\mathbf{s}$, as HPYLM does.

Since $p(n|h)$ has a product form as (3), we can also write the above expression recursively by introducing an auxiliary probability $p(s|h, n^+)$ as follows:

$$p(s|h, n^+) = q_n \cdot p(s|h, n) + (1 - q_n) \cdot p(s|h, (n+1)^+), \tag{11}$$
$$p(s|h) \equiv p(s|h, 0^+). \tag{12}$$

This formula shows that $q_n$ in fact defines the stick-breaking process on an infinite tree, where breaking proportions will differ branch to branch as opposed to a single proportion on a unit interval used in ordinary Dirichlet processes. In practice, we can truncate the infinite recursion in (11) and rescale it to make $p(n|h)$ a proper distribution.

### 3.4  "Stochastic Phrases" on Suffix Tree

In the expression (9) above, $p(s, n|h)$ is the probability that the symbol $s$ is generated by a Markov process of order $n$ on $h$, that is, using the last $n$ symbols of $h$ as a Markov state. This means that a subsequence $s_{-n} \cdots s_{-1} s$ forms a "phrase": for example, when "Gaussians" was generated using a context "mixture of", we can consider "mixture of Gaussians" as a phrase and assign a probability to this subsequence, which represents its cohesion strength irrespective of its length.

In other words, instead of emitting a single symbol $s$ at the root node of suffix tree, we can first stochastically descend the tree according to the probability to stop by (3). Finally, we emit $s$ given the context $s_{-n} \cdots s_{-1}$, which yields a phrase $s_{-n} \cdots s_{-1} s$ and its cohesion probability. Therefore, by traversing the suffix tree, we can compute $p(s, n|h)$ for all the subsequences efficiently. For concrete examples, see Figure 8 and 10 in Section 4.

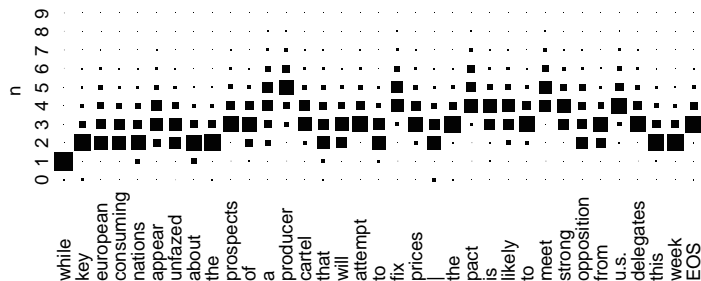

Figure 6: Estimated Markov order distributions from which each word has been generated.

## 4 Experiments

To investigate the behavior of the infinite Markov model, we conducted experiments on character and word sequences in natural language.

### 4.1 Infinite character Markov model

Character-based Markov model is widely employed in data compression and has important application in language processing, such as OCR and unknown word recognition. In this experiment, we used a 140,931 characters text of "Alice in Wonderland" and built an infinite Markov model using uniform Beta prior and truncation threshold $\epsilon = 0.0001$ in Section 3.2.

| Max. order | Perplexity |
|---|---|
| $n = 3$ | 6.048 |
| $n = 5$ | 3.803 |
| $n = 10$ | 3.519 |
| $n = \infty$ | **3.502** |

Table 1: Perplexity results of Character models.

Figure 5(a) is a random walk generation from this infinite model. To generate this, we begin with an infinite sequence of 'beginning of sentence' special symbols, and sample the next character according to the generative model given the already sampled sequence as the context. Figure 5(b) is the actual Markov orders used for generation by (8). Without any notion of "word", we can see that our model correctly captures it and even higher dependencies between "words". In fact, the model contained many nodes that correspond to valid words as well as the connective fragments between them. Table 1 shows predictive perplexity[4] results on separate test data. Compared with truncations $n = 3, 5$ and 10, the infinite model performs the best in all the variable order options.

### 4.2 Bayesian $\infty$-gram Language Model

**Data** For a word-based "n-gram" model of language, we used a random subset of the standard NAB Wall Street Journal language modeling corpus [12] [5], totalling 10,007,108 words (409,246 sentences) for training and 10,000 sentences for testing. Symbols that occurred fewer than 10 times in total and punctuation (commas, quotation marks etc.) are mapped to special characters, and all sentences are lowercased, yielding a lexicon of 26,497 words. As HPYLM is shown to converge very fast [9], according to preliminary experiments we used $N = 200$ Gibbs iterations for burn-in, and a further 50 iterations to evaluate the perplexity of the test data.

**Results** Figure 6 shows the Hinton diagram of estimated Markov order distributions on part of the training data, computed according to (7). As for the perplexity, Table 2 shows the results compared with the fixed-order HPYLM with the number of nodes in each model. $n$ means the fixed order for HPYLM, and the maximum order $n_{\max}$ in VPYLM. For the "infinite" model of $n = \infty$, we used a threshold $\epsilon = 10^{-8}$ in Section 3.2 for descending the suffix tree.

As empirically found by [12], perplexities will saturate when $n$ becomes large, because only a small portion of words actually exhibit long-range dependencies. However, we can see that the VPYLM performance is comparable to that of HPYLM with much fewer nodes and restaurants up to $n = 7$ and 8, where vanilla HPYLM encounters memory overflow caused by a rapid increase in the number of parameters. In fact, the inference of VPYLM is about 20% faster than that of HPYLM of the

| $n$ | HPYLM | VPYLM | Nodes(H) | Nodes(V) |
|---|---|---|---|---|
| 3 | 113.60 | 113.74 | 1,417K | **1,344K** |
| 5 | 101.08 | 101.69 | 12,699K | **7,466K** |
| 7 | N/A | **100.68** | *27,193K* | 10,182K |
| 8 | N/A | **100.58** | *34,459K* | 10,434K |
| $\infty$ | — | **100.36** | — | **10,629K** |

Table 2: Perplexity Results of VPYLM and HPYLM on the NAB corpus with the number of nodes in each model. N/A means a memory overflow caused by the expected number of nodes shown in italic.

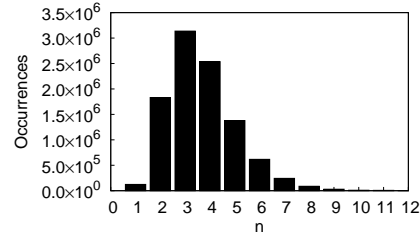

Figure 7: Global distribution of sampled Markov orders on the $\infty$-gram VPYLM over the NAB corpus. $n = 0$ is unigram, $n = 1$ is bigram,$\cdots$.

same order despite the additional cost of sampling n-gram orders, because it appropriately avoids the addition of unnecessarily deep nodes on the suffix tree. The perplexity at $n = \infty$ is the lowest compared to all fixed truncations, and contains only necessary number of nodes in the model.

Figure 7 shows a global n-gram order distribution from a single posterior sample of Gibbs iteration in $\infty$-gram VPYLM. Note that since we added an infinite number of dummy symbols to the sentence heads as usual, every word context has a maximum possible length of $\infty$. We can see from this figure that the context lengths that were actually used decay largely exponentially, as intuitively expected. Because of the tradeoff between using a longer, more predictive context and the penalty incurred when reaching a deeper node, interestingly a peak emerges around $n = 3 \sim 4$ as a global phenomenon.

With regard to the hyperparameter that defines the prior forms of suffix trees, we used a $(4, 1)$-prior in this experiment. In fact, this hyperparameter can be optimized by the empirical Bayes method using each Beta posterior of $q_i$ in (6). By using the Newton-Raphson iteration of [13], this converged to $(0.85, 0.57)$ on a 1 million word subset of the NAB corpus. However, we can see that the performance does not depend significantly on the prior. Figure 9 shows perplexity results for the same data, using $(\alpha, \beta) \in (0.1 \sim 10) \times (0.1 \sim 10)$. We can see from this figure that the performance is almost stable, except when $\beta$ is significantly greater than $\alpha$. Finally, we show in Figure 8 some "stochastic pharases" in Section 3.4 induced on the NAB corpus.

## 4.3 Variable Order Topic Model

While previous approaches to latent topic modeling assumed a fixed order such as unigrams or bigrams, the order is generally not fixed and unknown to us. Therefore, we used a Gibbs sampler for the Markov chain LDA [14] and augmented it by sampling Markov orders at the same time.

Because "topic-specific" sequences constitute only some part of the entire data, we assumed that the "generic" model generated the document according to probability $\lambda$, and the rest are generated by the LDA of VPYLM. We endow $\lambda$ a uniform Beta prior and used the posterior estimate for sampling that will differ document to document.

For the experiment, we used the NIPS papers dataset of 1739 documents. Among them, we used random 1500 documents for training and random 50 documents from the rest of 239 documents for testing, after the same preprocessing for the NAB corpus. We set a symmetric Dirichlet prior

| $p(s, n)$ | Stochastic phrases in the suffix tree |
|---|---|
| 0.9784 | primary new issues |
| 0.9726 | ^ at the same time |
| 0.9512 | is a unit of |
| 0.9026 | from # % in # to # % |
| 0.8896 | in a number of |
| 0.8831 | in new york stock exchange composite trading |
| 0.7566 | mechanism of the european monetary |
| 0.7134 | increase as a result of |
| 0.6617 | tiffany & co. |
| : | |

Figure 8: "Stochastic phrases" induced by the 8-gram VPYLM trained on the NAB corpus.

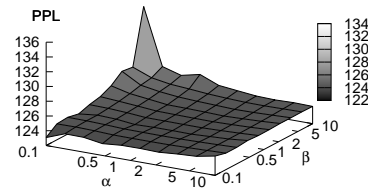

Figure 9: Perplexity results using different hyperparameters on the 1M NAB corpus.

| $p(n,s)$ | Phrase | $p(n,s)$ | Phrase | $p(n,s)$ | Phrase |
|---|---|---|---|---|---|
| 0.9904 | in section # | 0.9853 | et al | 0.9823 | monte carlo |
| 0.9900 | the number of | 0.9840 | receptive field | 0.9524 | associative memory |
| 0.9856 | in order to | 0.9630 | excitatory and inhibitory | 0.9081 | as can be seen |
| 0.9832 | in table # | 0.9266 | in order to | 0.8206 | parzen windows |
| 0.9752 | dealing with | 0.8939 | primary visual cortex | 0.8044 | in the previous section |
| 0.9693 | with respect to | 0.8756 | corresponds to | 0.7790 | american institute of physics |
| (a) Topic 0 ("generic") | | (b) Topic 1 | | (c) Topic 4 | |

Figure 10: Topic based stochastic pharases.

$\gamma = 0.1$ and the number of topics $M = 5$, $n_{max} = 5$ and ran a $N = 200$ Gibbs iterations to obtain a single posterior set of models.

Although in predictive perplexity the improvements are slight (VPYLDA=116.62, VPYLM=117.28), "stochastic pharases" computed on each topic VPYLM show interesting characteristics shown in Figure 10. Although we used a small number of latent topics in this experiment to avoid data sparsenesses, in future research we need a more flexible topic model where the number of latent topics will differ from node to node in the suffix tree.

## 5    Discussion and Conclusion

In this paper, we presented a completely generative approach to estimating variable order Markov processes. By extending a stick-breaking process "vertically" over a suffix tree of hierarchical Chinese restaurant processes, we can make a posterior inference on the Markov orders from which each data originates.

Although our architecture looks similar to Polya Trees [15], in Polya Trees their recursive partitions are independent while our stick-breakings are hierarchically organized according to the suffix tree. In addition to apparent application of our approach to hierarchical continuous distributions like Gaussians, we expect that the basic model can be used for the distribution of latent variables. Each data is assigned to a deeper level just when needed, and resides not only in leaf nodes but also in the intermediate nodes, by stochastically descending a clustering hierarchy from the root as described in this paper.

## Footnotes

[1]This is the leftmost path in Figure 1(a). When there is no corresponding branch, we will create it.

[2]This is a specific application of our model to the hierarchical Pitman-Yor processes for discrete data.

[3]Notice that by setting $(\alpha, \beta) = (0, \infty)$, we always obtain $q_i = 0$: with some maximum order $n_{\max}$, this is equivalent to always using the maximum depth, and thus to reducing the model to the original HPYLM. In this regard, VPYLM is a natural superset that includes HPYLM [9].

[4]Perplexity is a reciprocal of average predictive probabilities, thus smaller is better.

[5]We also conducted experiments on standard corpora of Chinese (character-wise) and Japanese, and obtained the same line of results presented in this paper.

## References

[1] C. E. Shannon. A mathematical theory of communication. *Bell System Technical Journal*, 27:379–423, 623–656, 1948.

[2] Alberto Apostolico and Gill Bejerano. Optimal amnesic probabilistic automata, or, how to learn and classify proteins in linear time and space. *Journal of Computational Biology*, 7:381–393, 2000.

[3] F.M.J. Willems, Y.M. Shtarkov, and T.J. Tjalkens. The Context-Tree Weighting Method: Basic Properties. *IEEE Trans. on Information Theory*, 41:653–664, 1995.

[4] Frederick Jelinek. *Statistical Methods for Speech Recognition*. Language, Speech, and Communication Series. MIT Press, 1998.

[5] Peter Buhlmann and Abraham J. Wyner. Variable Length Markov Chains. *The Annals of Statistics*, 27(2):480–513, 1999.

[6] Fernando Pereira, Yoram Singer, and Naftali Tishby. Beyond Word N-grams. In *Proc. of the Third Workshop on Very Large Corpora*, pages 95–106, 1995.

[7] Dana Ron, Yoram Singer, and Naftali Tishby. The Power of Amnesia. In *Advances in Neural Information Processing Systems*, volume 6, pages 176–183, 1994.

[8] Andreas Stolcke. Entropy-based Pruning of Backoff Language Models. In *Proc. of DARPA Broadcast News Transcription and Understanding Workshop*, pages 270–274, 1998.

[9] Yee Whye Teh. A Bayesian Interpretation of Interpolated Kneser-Ney. Technical Report TRA2/06, School of Computing, NUS, 2006.

[10] Sharon Goldwater, Thomas L. Griffiths, and Mark Johnson. Interpolating Between Types and Tokens by Estimating Power-Law Generators. In *NIPS 2005*, 2005.

[11] Daniel Sleator and Robert Tarjan. Self-Adjusting Binary Search Trees. *JACM*, 32(3):652–686, 1985.

[12] Joshua T. Goodman. A Bit of Progress in Language Modeling, Extended Version. Technical Report MSR–TR–2001–72, Microsoft Research, 2001.

[13] Thomas P. Minka. Estimating a Dirichlet distribution, 2000. http://research.microsoft.com/~minka/papers/dirichlet/.

[14] Mark Girolami and Ata Kabán. Simplicial Mixtures of Markov Chains: Distributed Modelling of Dynamic User Profiles. In *NIPS 2003*. 2003.

[15] R. Daniel Mauldin, William D. Sudderth, and S. C. Williams. Polya Trees and Random Distributions. *Annals of Statistics*, 20(3):1203–1221, 1992.
